# Position Variance, Recurrence and Perceptual Learning

**Zhaoping Li**      **Peter Dayan**
Gatsby Computational Neuroscience Unit
17 Queen Square, London, England, WC1N 3AR.
zhaoping@gatsby.ucl.ac.uk          dayan@gatsby.ucl.ac.uk

## Abstract

Stimulus arrays are inevitably presented at different positions on the retina in visual tasks, even those that nominally require fixation. In particular, this applies to many perceptual learning tasks. We show that perceptual inference or discrimination in the face of positional variance has a structurally different quality from inference about fixed position stimuli, involving a particular, quadratic, non-linearity rather than a purely linear discrimination. We show the advantage taking this non-linearity into account has for discrimination, and suggest it as a role for recurrent connections in area V1, by demonstrating the superior discrimination performance of a recurrent network. We propose that learning the feedforward and recurrent neural connections for these tasks corresponds to the fast and slow components of learning observed in perceptual learning tasks.

## 1 Introduction

The field of perceptual learning in simple, but high precision, visual tasks (such as vernier acuity tasks) has produced many surprising results whose import for models has yet to be fully felt. A core of results is that there are two stages of learning, one fast, which happens over the first few trials, and another slow, which happens over multiple sessions, may involve REM sleep, and can last for months or even years (Fahle, 1994; Karni & Sagi, 1993; Fahle, Edelman, & Poggio 1995). Learning is surprisingly specific, in some cases being tied to the eye of origin of the input and rarely admitting generalisation across wide areas of space or between tasks that appear extremely similar, even involving the same early-stage detectors (*eg* Fahle, Edelman, & Poggio 1995; Fahle, 1994). For instance, improvement through learning on an orientation discrimination task does not lead to improvement on a vernier acuity task (Fahle 1997), even though both tasks presumably use the same orientation selective striate cortical cells to process inputs.

Of course, learning in human psychophysics is likely to involve plasticity in a large number of different parts of the brain over various timescales. Previous studies (Poggio, Fahle, & Edelman 1992, Weiss, Edelman, & Fahle 1993) proposed phenomenological models of learning in a feedforward network architecture. In these models, the first stage units in the network receive the sensory inputs through the medium of basis functions relevant for the perceptual task. Over learning, a set of feedforward weights is acquired such that the weighted sum of the activities from the input units can be used to make an appropriate binary decision, *eg* using a threshold. These models can account for some, but not all, observations on perceptual learning (Fahle et al 1995). Since the activity of V1 units seems not to relate directly to behavioral decisions on these visual tasks, the feedforward connections

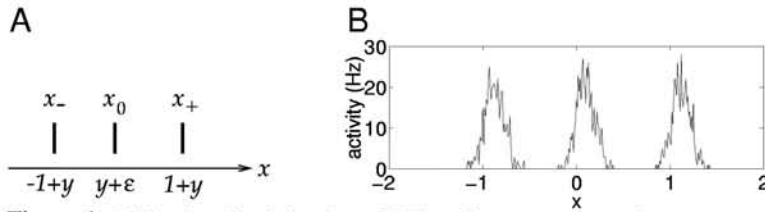

Figure 1: Mid-point discrimination. A) Three bars are presented at $x_-$, $x_0$ and $x_+$. The task is to report which of the outer bars is closer to the central bar. $y$ represents the variable placement of the stimulus array. B) Population activities in cortical cells evoked by the stimulus bars — the activities $a_i$ is plotted against the preferred location $x_i$ of the cells. This comes from Gaussian tuning curves ($k = 20; \tau = 0.1$) and Poisson noise. There are 81 units whose preferred values are placed at regular intervals of $\Delta x = 0.05$ between $x = -2$ and $x = 2$.

must model processing beyond V1. The lack of generalisation between tasks that involve the same visual feature samplers suggests that the basis functions, *eg* the orientation selective primary cortical cells that sample the inputs, do not change their sensitivity and shapes, *eg* their orientation selectivity or tuning widths. However, evidence such as the specificity of learning to the eye of origin and spatial location strongly suggest that lower visual areas such as V1 are directly involved in learning. Indeed, V1 is a visual processor of quite some computational power (performing tasks such as segmentation, contour-integration, pop-out, noise removal) rather than being just a feedforward, linear, processing stage (*eg* Li, 1999; Pouget *et al* 1998).

Here, we study a paradigmatic perceptual task from a statistical perspective. Rather than suggest particular learning rules, we seek to understand what it is about the structure of the task that might lead to two phases of learning (fast and slow), and thus what computational job might be ascribed to V1 processing, in particular, the role of lateral recurrent connections. We agree with the general consensus that fast learning involves the feedforward connections. However, by considering positional invariance for discrimination, we show that there is an inherently non-linear component to the overall task, which defeats feedforward algorithms.

## 2 The bisection task

Figure 1A shows the bisection task. Three bars are presented at horizontal positions $x_0 = y + \epsilon$, $x_- = -1 + y$ and $x_+ = 1 + y$, where $-1 \ll \epsilon \ll 1$. Here $y$ is a *nuisance* random number with zero mean, reflecting the variability in the position of stimulus array due to eye movements or other uncontrolled factors. The task for the subject is to report which of the outer bars is closer to the central bar, *ie* to report whether $\epsilon$ is greater than or less than 0. The bars create a population-coded representation in V1 cells preferring vertical orientation. In figure 1B, we show the activity of cells $a_i$ as a function of preferred topographic location $x_i$ of the cell; and, for simplicity, we ignore activities from other V1 cells which prefer orientations other than vertical.

We assume that the cortical response to the bars is additive, with mean

$$\bar{a}_i(\epsilon, y) = f(x_i - x_0) + f(x_i - x_-) + f(x_i - x_+) \tag{1}$$

(we often drop the dependence on $\epsilon, y$ and write $\bar{a}_i$, or, for all the components, $\bar{\mathbf{a}}$) where $f$ is, say, a Gaussian, tuning curve with height $k$ and tuning width $\tau$, $f(x) = ke^{-x^2/2\tau^2}$, usually with $\tau \ll 1$. The net activity is $a_i = \bar{a}_i + n_i$, where $n_i$ is a noise term. We assume that $n_i$ comes from a Poisson distribution and is independent across the units, and $\epsilon$ and $y$ have mean zero and are uniformly distributed in their respective ranges.

The subject must report whether $\epsilon$ is greater or less than 0 on the basis of the activities $\mathbf{a}$.

A normative way to do this is to calculate the probability $P[\epsilon|\mathbf{a}]$ of $\epsilon$ given $\mathbf{a}$, and report by maximum likelihood (ML) that $\epsilon > 0$ if $\int_{\epsilon>0} d\epsilon\, P[\epsilon|\mathbf{a}] > 0.5$. Without prior information about $\epsilon, y$, and with Poisson noise $n_i = a_i - \bar{a}_i$, we have

$$P[\epsilon, y|\mathbf{a}] = P[\mathbf{a}|\epsilon, y]P[\epsilon, y]/P(\mathbf{a}) \propto P[\mathbf{a}|\epsilon, y] \propto \prod_i e^{-\bar{a}_i(\epsilon,y)}(\bar{a}_i(\epsilon, y))^{a_i} \qquad (2)$$

## 3   Fixed position stimulus array

When the stimulus array is in a fixed position $y = 0$, analysis is easy, and is very similar to that carried out by Seung & Sompolinsky (1993). Dropping $y$, we calculate $\log P[\epsilon|\mathbf{a}]$ and approximate it by Taylor expansion about $\epsilon = 0$ to second order in $\epsilon$:

$$\log P[\mathbf{a}|\epsilon] \sim \text{constant} + \epsilon \left.\tfrac{\partial}{\partial \epsilon}\log P[\mathbf{a}|\epsilon]\right|_{\epsilon=0} + \tfrac{\epsilon^2}{2}\left.\tfrac{\partial^2}{\partial \epsilon^2}\log P[\mathbf{a}|\epsilon]\right|_{\epsilon=0} \qquad (3)$$

ignoring higher order terms. Provided that the last term is negative (which it indeed is, almost surely), we derive an approximately Gaussian distribution

$$P[\epsilon|\mathbf{a}] \propto \exp[-(\epsilon - \bar{\epsilon})^2/(2\sigma_\epsilon^2)] \qquad (4)$$

with variance $\sigma_\epsilon^2 \equiv [-\left.\tfrac{\partial^2}{\partial \epsilon^2}\log P[\mathbf{a}|\epsilon]\right|_{\epsilon=0}]^{-1}$ and mean $\bar{\epsilon} \equiv \sigma_\epsilon^2 \tfrac{\partial}{\partial \epsilon}\log P[\mathbf{a}|\epsilon]|_{\epsilon=0}$. Thus the subject should report that $\epsilon > 0$ or $\epsilon < 0$ if the test $t(\mathbf{a}) = \tfrac{\partial}{\partial \epsilon}\log P[\mathbf{a}|\epsilon]|_{\epsilon=0}$ is greater or less than zero respectively. For the Poisson noise case we consider, $\log P[\mathbf{a}|\epsilon] = \text{constant} + \sum_i a_i \log \bar{a}_i(\epsilon)$ since $\sum_i \bar{a}_i(\epsilon)$ is a constant, independent of $\epsilon$. Thus,

$$t(\mathbf{a}) = \sum_i a_i \left.\tfrac{\partial}{\partial \epsilon}\log \bar{a}_i(\epsilon)\right|_{\epsilon=0} \qquad (5)$$

Therefore, maximum likelihood discrimination can be implemented by a linear feedforward network mapping inputs $a_i$ through feedforward weights $w_i = \tfrac{\partial}{\partial \epsilon}\log \bar{a}_i$ to calculate as the output $t(\mathbf{a}) = \sum_i w_i a_i$. A threshold of 0 on $t(\mathbf{a})$ provides the discrimination $\epsilon > 0$ if $t(\mathbf{a}) > 0$ and $\epsilon < 0$ for $t(\mathbf{a}) < 0$. The task therefore has an essentially *linear character*. Note that if the noise corrupting the activities is Gaussian, the weights should instead be $w_i = \tfrac{\partial}{\partial \epsilon}\bar{a}_i$.

Figure 2A shows the optimal discrimination weights for the case of independent Poisson noise. The lower solid line in figure 2C shows optimal performance as a function of $\epsilon$. The error rate drops precipitately from 50% for very small (and thus difficult) $\epsilon$ to almost 0, long before $\epsilon$ approaches the tuning width $\tau$.

It is also possible to learn weights in a variety of ways (*eg* Poggio, Fahle & Edelman, 1992; Weiss, Edelman & Fahle, 1993; Fahle, Edelman & Poggio 1995;) Figure 2B shows discrimination weights learned using a simple error-correcting learning procedure, which are almost the same as the optimal weights and lead to performance that is essentially optimal (the lower dashed line in figure 2C) . We use error-correcting learning as a comparison technique below.

## 4   Moveable stimulus array

If the stimulus array can move around, *ie* if $y$ is not necessarily 0, then the discrimination task gets considerably harder. The upper dotted line in figure 2C shows the (rather unfair) test of using the learned weights in figure 2B when $y \in [-.2, .2]$ varies uniformly. Clearly this has a highly detrimental effect on the quality of discrimination. Looking at the weight structure in figure 2A;B suggests an obvious reason for this – the weights associated with the outer bars are zero since they provide no information about $\epsilon$ when $y = 0$, and the

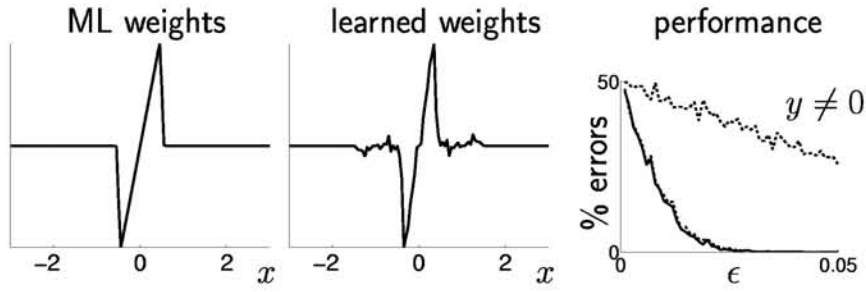

Figure 2: A) The ML optimal discrimination weights $\mathbf{w} = \frac{\partial}{\partial\epsilon}\log\bar{\mathbf{a}}$ (plotted as $w_i$ vs. $x_i$) for deciding if $\epsilon > 0$ when $y = 0$. B) The learned discrimination weights $\mathbf{w}$ for the same decision. During on line learning, random examples were selected with $\epsilon \in -2[-\tau,\tau]$ uniformly, $\tau = 0.1$, and the weights were adjusted online to maximise the log probability of generating the correct discrimination under a model in which the probability of declaring that $\epsilon > 0$ is $\sigma(\sum_i w_i a_i) = 1/(1 + \exp(-\sum_i w_i a_i))$. C) Performance of the networks with ML (lower solid line) and learned (lower dashed line) weights as a function of $\epsilon$. Performance is measured by drawing $\mathbf{a}$ randomly given $\epsilon$ and $y$, and assessing the %'age of trials the answer is incorrect. The upper dotted line shows the effect of drawing $y \in [-0.2, 0.2]$ uniformly, yet using the ML weights in (B) that assume $y = 0$.

weights are finely balanced about 0, the mid-point of the outer bars, giving an unbiased or balanced discrimination on $\epsilon$. If the whole array can move, this balance will be destroyed, and all the above conclusions change.

The equivalent of equation (3) when $y \neq 0$ is

$$\log P[\epsilon, y | \mathbf{a}] \sim \text{constant} + \left(\epsilon\frac{\partial}{\partial\epsilon} + y\frac{\partial}{\partial y} + \frac{\epsilon^2}{2}\frac{\partial^2}{\partial\epsilon^2} + \frac{y^2}{2}\frac{\partial^2}{\partial y^2} + \epsilon y\frac{\partial^2}{\partial y\partial\epsilon}\right)\log P[\mathbf{a}|\epsilon, y]|_{\epsilon, y = 0}$$

Thus, to second-order, a Gaussian distribution can approximate $P[\epsilon, y|\mathbf{a}]$. Figure 3A shows the high quality of this approximation. Here, $\epsilon$ and $y$ are anti-correlated given activities $\mathbf{a}$, because the information from the center stimulus bar only constrains their sum $\epsilon + y$. Of interest is the probability $P[\epsilon|\mathbf{a}] = \int dy \log P[\epsilon, y|\mathbf{a}]$, which is approximately Gaussian with mean $\beta\rho_\epsilon^2$ and variance $\rho_\epsilon^2$, where, under Poisson noise $n_i = a_i - \bar{a}_i$,

$$\beta = [\mathbf{a} \cdot \tfrac{\partial\log\bar{\mathbf{a}}}{\partial\epsilon} - (\mathbf{a} \cdot \tfrac{\partial^2\log\bar{\mathbf{a}}}{\partial y\partial\epsilon})(\mathbf{a} \cdot \tfrac{\partial\log\bar{\mathbf{a}}}{\partial y})/(\mathbf{a} \cdot \tfrac{\partial^2\log\bar{\mathbf{a}}}{\partial y^2})]|_{\epsilon, y = 0}$$

$$\rho_\epsilon^{-2} = [(\mathbf{a} \cdot \tfrac{\partial^2\log\bar{\mathbf{a}}}{\partial y\partial\epsilon})^2/(\mathbf{a} \cdot \tfrac{\partial^2\log\bar{\mathbf{a}}}{\partial y^2}) - \mathbf{a} \cdot \tfrac{\partial^2\log\bar{\mathbf{a}}}{\partial\epsilon^2}]|_{\epsilon, y = 0}$$

Since $-\mathbf{a} \cdot \frac{\partial^2\log\bar{\mathbf{a}}}{\partial y^2}$ (which is the inverse variance of the Gaussian distribution of $y$ that we integrated out) is positive, the appropriate test for the sign of $\epsilon$ is

$$t(\mathbf{a}) = [(\mathbf{a} \cdot \tfrac{\partial^2\log\bar{\mathbf{a}}}{\partial y\partial\epsilon})(\mathbf{a} \cdot \tfrac{\partial\log\bar{\mathbf{a}}}{\partial y}) - (\mathbf{a} \cdot \tfrac{\partial\log\bar{\mathbf{a}}}{\partial\epsilon})(\mathbf{a} \cdot \tfrac{\partial^2\log\bar{\mathbf{a}}}{\partial y^2})]|_{\epsilon, y = 0} \qquad (6)$$

If $t(\mathbf{a}) > 0$ then we should report $\epsilon > 0$, and conversely. Interestingly, $t(\mathbf{a})$ is a very simple quadratic form

$$t(\mathbf{a}) = \mathbf{a} \cdot \mathbf{Q} \cdot \mathbf{a} \equiv \sum_{ij} a_i a_j \left[(\tfrac{\partial^2\log\bar{a}_i}{\partial y\partial\epsilon})(\tfrac{\partial\log\bar{a}_j}{\partial y}) - (\tfrac{\partial\log\bar{a}_i}{\partial\epsilon})(\tfrac{\partial^2\log\bar{a}_j}{\partial y^2})\right]|_{\epsilon, y = 0} \qquad (7)$$

Therefore, the discrimination problem in the face of positional variance has a precisely quantifiable *non-linear* character. The quadratic test $t(\mathbf{a})$ cannot be implemented by a linear feedforward architecture only, since the optimal boundary $t(\mathbf{a}) = 0$ to separate the state space $\mathbf{a}$ for a decision is now curved. Writing $t(\mathbf{a}) = \mathbf{a} \cdot \mathbf{Q} \cdot \mathbf{a}$ where the symmetric

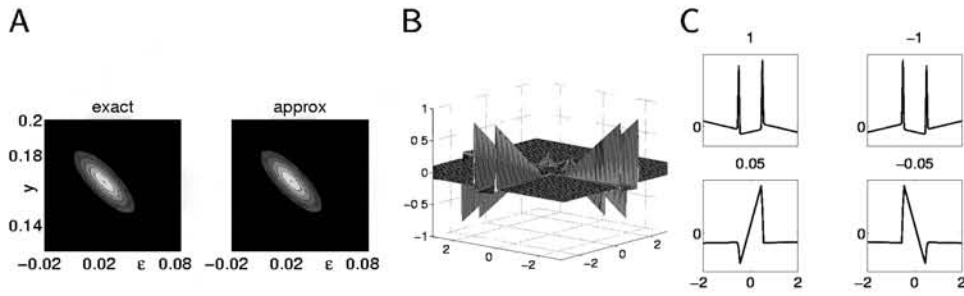

Figure 3: Varying $y$. A) Posterior distribution $P[\epsilon, y|\mathbf{a}]$. Exact (left) $P[\epsilon, y|\mathbf{a}]$ for a particular $\mathbf{a}$ with true values $\epsilon = 0.2\tau$, $y = 1.5\tau$ (with $\tau = 0.1$) and its bivariate Gaussian approximation (right). Only the relevant region of $(\epsilon, y)$ space is shown – outside this, the probability mass is essentially 0 (and the contour values are the same). B) The quadratic form $\mathbf{Q}$, $Q_{ij}$ vs. $x_i$ and $x_j$. C) The four eigenvectors of $\mathbf{Q}$ with non-zero eigenvalues (shown above). The eigenvalues come in $\pm$ pairs; the associated eigenvectors come in antisymmetric pairs. The absolute scale of $\mathbf{Q}$ and its eigenvalues is arbitrary.

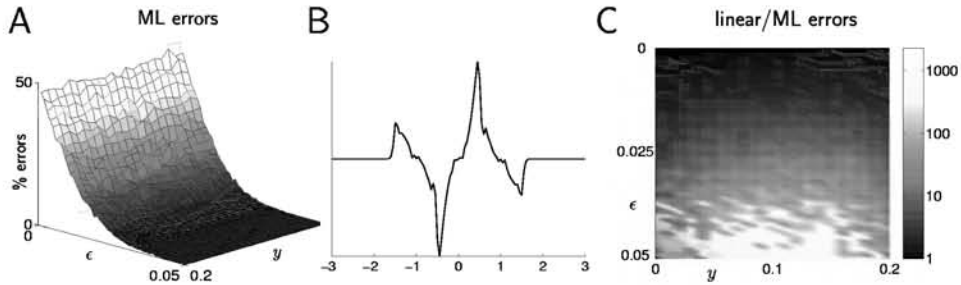

Figure 4: $y \neq 0$. A) Performance of the approximate inference based on the quadratic form of figure 3B in terms of %'age error as a function of $|y|$ and $|\epsilon|$ ($\tau = 0.1$). B) Feedforward weights, $w_i$ vs. $x_i$, learned using the same procedure as in figure 2B, but with $y \in [-.2, .2]$ chosen uniformly at random. C) Ratio of error rates for the linear (weights from B) to the quadratic discrimination. Values that would be infinite are pegged at 20.

form $Q_{ij} = (Q'_{ij} + Q'_{ji})/2$, we find $\mathbf{Q}$ only has four non-zero eigenvalues, for the 4-dimensional sub-space spanned by 4 vectors $\frac{\partial^2 \log \bar{\mathbf{a}}}{\partial y \partial \epsilon}|_{\epsilon,y=0}$, $\frac{\partial \log \bar{\mathbf{a}}}{\partial y}|_{\epsilon,y=0}$, $\frac{\partial \log \bar{\mathbf{a}}}{\partial \epsilon}|_{\epsilon,y=0}$, and $\frac{\partial^2 \log \bar{\mathbf{a}}}{\partial y^2}|_{\epsilon,y=0}$. $\mathbf{Q}$ and its eigenvectors and eigenvalues are shown in Figure 3B;C. Note that if Gaussian rather than Poisson noise is used for $n_i = a_i - \bar{a}_i$, the test $t(\mathbf{a})$ is still quadratic.

Using $t(\mathbf{a})$ to infer $\epsilon$ is sound for $y$ up to two standard deviations ($\tau$) of the tuning curve $f(x)$ away from 0, as shown in Figure 4A. By comparison, a feedforward network, of weights shown in figure 4B and learned using the same error-correcting learning procedure as above, gives substantially worse performance, even though it is better than the feedforward net of Figure 2A;B. Figure 4C shows the *ratio* of the error rates for the linear to the quadratic decisions. The linear network is often dramatically worse, because it fails to take proper account of $y$.

We originally suggested that recurrent interactions in the form of horizontal intra-cortical connections within V1 might be the site of the longer term improvement in behavior. Figure 5 demonstrates the plausibility of this idea. Input activity (as in figure 1B) is used to initialise the state $\mathbf{u}$ at time $t = 0$ of a recurrent network. The recurrent weights are

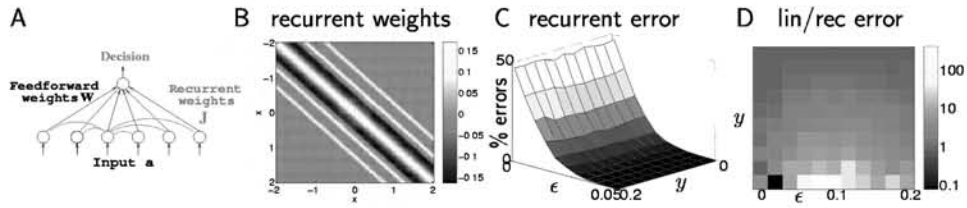

**Figure 5:** Threshold linear recurrent network, its weights, and performance. See text.

symmetric and shown in figure 5B. The network activities evolve according to

$$du_i/dt = -u_i + \sum_j J_{ij} g(u_j) + a_i \tag{8}$$

where $J_{ij}$ are the recurrent weight from unit $j$ to $i$, $g(u) = u$ if $u > 0$ and $g(u) = 0$ for $u \leq 0$. The network activities finally settle to an equilibrium $\mathbf{u}(t \to \infty)$ (note that $u_i(t \to \infty) = a_i$ when $J = 0$). The activity values $\mathbf{u}(t \to \infty)$ of this equilibrium are fed through feedforward weights $\mathbf{w}$, that are trained for this recurrent network just as for the pure feedforward case, to reach a decision $\sum_i w_i u_i(t \to \infty)$. Figure 5C shows that using this network gives results that are almost invariant to $y$ (as for the quadratic discriminator); and figure 5D shows that it generally outperforms the optimal linear discriminator by a large margin, albeit performing slightly worse than the quadratic form. The recurrent weight matrix is subject to three influences: (1) a short range interaction $J_{ij}$ for $|x_i - x_j| \lesssim \tau$ to stablize activities $a_i$ induced by a single bar in the input; (2) a longer range interaction $J_{ij}$ for $|x_i - x_j| \sim 1$ to mediate interaction between neighboring stimulus bars, amplifying the effects of the displacement signal $\epsilon$, and (3) a slight local interaction $J_{ij}$ for $|x_i|, |x_j| \lesssim \tau$. The first two interaction components are translation invariant in the spatial range of $x_i, x_j \in [-2, 2]$ where the stimulus array appears, in order to accommodate the positional variance in $y$. The last component is not translation invariant and counters variations in $y$.

## 5 Discussion

The problem of position invariant discrimination is common to many perceptual learning tasks, including hyper-acuity tasks such as the standard line vernier, three-dot vernier, curvature vernier, and orientation vernier tasks (Fahle et al 1995, Fahle 1997). Hence, the issues we address and analyze here are of general relevance. In particular, our mathematical formulation, derivations, and thus conclusions, are general and do not depend on any particular aspect of the bisection task. One essential problem in many of these tasks is to discriminate a stimulus variable $\epsilon$ that depends only on the relative positions between the stimulus features, while the absolute position $y$ of the whole stimulus array can vary between trials by an amount that is much larger than the discrimination threshold (or acuity) on $\epsilon$. The positional variable $y$ may not have to correspond to the absolute position of the stimulus array, but merely to the error in the estimation of the absolute position of the stimulus by other neural areas. Our study suggests that although when $y = 0$ is fixed, the discrimination is easy and soluble by a linear, feedforward network, whose weights can be learnt in a straight-forward manner, when $y$ is not fixed, optimal discrimination of $\epsilon$ is based on an approximately quadratic function of the input activities, which cannot be implemented using a linear feedforward net.

We also showed that a non-linear recurrent network, which is a close relative of a line attractor network, can perform much better than a pure feedforward network on the bisection task in the face of position variance. There is experimental evidence that lateral connections within V1 change after learning the bisection task (Gilbert 2000), although we have yet to construct an appropriate learning rule. We suggest that learning the recurrent weights for

the nonlinear transform corresponds to the slow component in perceptual learning, while learning the feedforward weights corresponds to the fast component. The desired recurrent weights are expected to be much more difficult to learn, in the face of nonlinear transforms and (the easily unstable) recurrent dynamics. Further, the feedforward weights need to be adjusted further as the recurrent weights change the activities on which they work.

The precise recurrent interactions in our network are very specific to the task and its parameters. In particular, the range of the interactions is completely determined by the scale of spacing between stimulus bars; and the distance-dependent excitation and inhibition in the recurrent weights is determined by the nature of the bisection task. This may be why there is little transfer of learning between tasks, when the nature and the spatial scale of the task change, even if the same input units are involved. However, our recurrent interaction model does predict that transfer is likely when the spacing between the two outer bars (here at $\Delta x = 2$) changes by a small fraction. Further, since the signs of the recurrent synapses change drastically with the distance between the interacting cells, negative transfer is likely between two bisection tasks of slightly different spatial scales. We are planning to test this prediction.

Achieving selectivity at the same time as translation invariance is a very basic requirement for position-invariant object recognition (see Riesenhuber & Poggio 1999 for a recent discussion), and arises in a pure form in this bisection task. Note, for instance, that trying to cope with different values of $y$ by *averaging* spatially shifted versions of the optimal weights for $y = 0$ (figure 2A) would be hopeless, since this would erase (or at very least blur) the precise spatial positioning of the peaks and troughs which underlies the discrimination power. It would be possible to *scan* the input for the value of $y$ that fits the best and then apply the discriminator centered about that value, and, indeed, this is conceptually what the neocognitron (Fukushima 1980) and the MAX-model (Riesenhuber & Poggio 1999) do using layers of linear and non-linear combination. In our case, we have shown, at least for fairly small $y$, that the optimal non-linearity for the task is a simple quadratic.

### Acknowledgements

Funding is from the Gatsby Charitable Foundation. We are very grateful to Shimon Edelman, Manfred Fahle and Maneesh Sahani for discussions.

# References

[1] Karni A and Sagi D. *Nature* **365** 250-252, 1993.

[2] Fahle M. Edelman S. and Poggio T. *Vision Res.* **35** 3003-3013, 1995.

[3] Fahle M. *Perception* **23** 411-427, (1994). And also Fahle M. *Vis. Res.* 37(14) 1885-1895, (1997).

[4] Poggio T. Fahle M. and Edelman S. *Science* **256** 1018-1021, 1992.

[5] Weiss Y. Edelman S. and Fahle M. *Neural Computation* **5** 695-718, 1993.

[6] Li, Zhaoping *Network: Computation in Neural Systems* 10(2) 187-212, 1999.

[7] Pouget A, Zhang K, Deneve S, Latham PE. *Neural Comput.* 10(2):373-401, 1998.

[8] Seung HS, Sompolinsky H. *Proc Natl Acad Sci U S A.* 90(22):10749-53, 1993

[9] Koch C. Biophysics of computation. Oxford University Press, 1999.

[10] Gilbert C. Presentation at the Neural Dynamics Workshop, Gatsby Unit, 2/2000.

[11] Riesenhuber M, Poggio T. *Nat Neurosci.* 2(11):1019-25, 1999.

[12] Fukushima, K. *Biol. Cybern.* **36** 193-202, 1980.
